# A Spectral Regularization Framework for Multi-Task Structure Learning

**Andreas Argyriou**
Department of Computer Science
University College London
Gower Street, London WC1E 6BT, UK
a.argyriou@cs.ucl.ac.uk

**Charles A. Micchelli**
Department of Mathematics and Statistics
SUNY Albany
1400 Washington Avenue
Albany, NY, 12222, USA

**Massimiliano Pontil**
Department of Computer Science
University College London
Gower Street, London WC1E 6BT, UK
m.pontil@cs.ucl.ac.uk

**Yiming Ying**
Department of Engineering Mathematics
University of Bristol
University Walk, Bristol, BS8 1TR, UK
enxyy@bristol.ac.uk

## Abstract

Learning the common structure shared by a set of supervised tasks is an important practical and theoretical problem. Knowledge of this structure may lead to better generalization performance on the tasks and may also facilitate learning new tasks. We propose a framework for solving this problem, which is based on regularization with *spectral functions* of matrices. This class of regularization problems exhibits appealing computational properties and can be optimized efficiently by an alternating minimization algorithm. In addition, we provide a necessary and sufficient condition for convexity of the regularizer. We analyze concrete examples of the framework, which are equivalent to regularization with $L_p$ matrix norms. Experiments on two real data sets indicate that the algorithm scales well with the number of tasks and improves on state of the art statistical performance.

## 1 Introduction

Recently, there has been renewed interest in the problem of multi-task learning, see [2, 4, 5, 14, 16, 19] and references therein. This problem is important in a variety of applications, ranging from conjoint analysis [12], to object detection in computer vision [18], to multiple microarray data set integration in computational biology [8] – to mention just a few. A key objective in many multi-task learning algorithms is to implement mechanisms for learning the possible structure underlying the tasks. Finding this common structure is important because it allows pooling information across the tasks, a property which is particularly appealing when there are many tasks but only few data per task. Moreover, knowledge of the common structure may facilitate learning new tasks (transfer learning), see [6] and references therein.

In this paper, we extend the formulation of [4], where the structure shared by the tasks is described by a positive definite matrix. In Section 2, we propose a framework in which the task parameters and the structure matrix are jointly computed by minimizing a regularization function. This function has the following appealing property. When the structure matrix is fixed, the function decomposes across the tasks, which can hence be learned independently with standard methods such as SVMs. When the task parameters are fixed, the optimal structure matrix is a *spectral function* of the covariance of the tasks and can often be explicitly computed. As we shall see, spectral functions are of particular interest in this context because they lead to an efficient alternating minimization algorithm.

The contribution of this paper is threefold. First, in Section 3 we provide a necessary and sufficient condition for convexity of the optimization problem. Second, in Section 4 we characterize the spectral functions which relate to Schatten $L_p$ regularization and present the alternating minimization algorithm. Third, in Section 5 we discuss the connection between our framework and the convex optimization method for learning the kernel [11, 15], which leads to a much simpler proof of the convexity in the kernel than the one given in [15]. Finally, in Section 6 we present experiments on two real data sets. The experiments indicate that the alternating algorithm runs significantly faster than gradient descent and that our method improves on state of the art statistical performance on these data sets. They also highlight that our approach can be used for transfer learning.

## 2 Modelling Tasks' Structure

In this section, we introduce our multi-task learning framework. We denote by $\mathbf{S}^d$ the set of $d \times d$ symmetric matrices, by $\mathbf{S}^d_+$ ($\mathbf{S}^d_{++}$) the subset of positive semidefinite (definite) ones and by $\mathbf{O}^d$ the set of $d \times d$ orthogonal matrices. For every positive integer $n$, we define $\mathbb{N}_n = \{1, \ldots, n\}$. We let $T$ be the number of tasks which we want to simultaneously learn. We assume for simplicity that each task $t \in \mathbb{N}_T$ is well described by a linear function defined, for every $x \in \mathbb{R}^d$, as $w_t^\top x$, where $w_t$ is a fixed vector of coefficients. For each task $t \in \mathbb{N}_T$, there are $m$ data examples $\{(x_{tj}, y_{tj}) : j \in \mathbb{N}_m\} \subset \mathbb{R}^d \times \mathbb{R}$ available. In practice, the number of examples per task may vary but we have kept it constant for simplicity of notation.

Our goal is to learn the vectors $w_1, \ldots, w_T$, as well as the common structure underlying the tasks, from the data examples. In this paper we follow the formulation in [4], where the tasks' structure is summarized by a positive definite matrix $D$ which is linked to the covariance matrix between the tasks, $WW^\top$. Here, $W$ denotes the $d \times T$ matrix whose $t$-th column is given by the vector $w_t$ (we have assumed for simplicity that the mean task is zero). Specifically, we learn $W$ and $D$ by minimizing the function

$$\text{Reg}(W, D) := \text{Err}(W) + \gamma \, \text{Penalty}(W, D), \tag{2.1}$$

where $\gamma$ is a positive parameter which balances the importance between the error and the penalty. The former may be any bounded from below and convex function evaluated at the values $w_t^\top x_{tj}$, $t \in \mathbb{N}_T$, $j \in \mathbb{N}_m$. Typically, it will be the average error on the tasks, namely, $\text{Err}(W) = \sum_{t \in \mathbb{N}_T} L_t(w_t)$, where $L_t(w_t) = \sum_{j \in \mathbb{N}_m} \ell(y_{tj}, w_t^\top x_{tj})$ and $\ell : \mathbb{R} \times \mathbb{R} \to [0, \infty)$ is a prescribed loss function (e.g. quadratic, SVM, logistic etc.). We shall assume that the loss $\ell$ is convex in its second argument, which ensures that the function $\text{Err}$ is also convex. The latter term favors the tasks sharing some common structure and is given by

$$\text{Penalty}(W, D) = \text{tr}(F(D)WW^\top) = \sum_{t=1}^T w_t^\top F(D) w_t, \tag{2.2}$$

where $F : \mathbf{S}^d_{++} \to \mathbf{S}^d_{++}$ is a prescribed *spectral* matrix function. This is to say that $F$ is induced by applying a function $f : (0, \infty) \to (0, \infty)$ to the eigenvalues of its argument. That is, for every $D \in \mathbf{S}^d_{++}$ we write $D = U\Lambda U^\top$, where $U \in \mathbf{O}^d$, $\Lambda = \text{Diag}(\lambda_1, \ldots, \lambda_d)$, and define

$$F(D) = U F(\Lambda) U^\top, \quad F(\Lambda) = \text{Diag}(f(\lambda_1), \ldots, f(\lambda_d)). \tag{2.3}$$

In the rest of the paper, we will always use $F$ to denote a spectral matrix function and $f$ to denote the associated real function, as above.

Minimization of the function $\text{Reg}$ allows us to learn the tasks and at the same time a good representation for them which is summarized by the eigenvectors and eigenvalues of the matrix $D$. Different choices of the function $f$ reflect different properties which we would like the tasks to share. In the special case that $f$ is a constant, the tasks are totally independent and the regularizer (2.2) is a sum of $T$ independent $L_2$ regularizers. In the case $f(\lambda) = \lambda^{-1}$, which is considered in [4], the regularizer favors a *sparse* representation in the sense that the tasks share a small common set of features. More generally, functions of the form $f(\lambda) = \lambda^{-\alpha}, \alpha \geq 0$, allow for combining shared features and task-specific features to some degree tuned by the exponent $\alpha$. Moreover, the regularizer (2.2) ensures that the optimal representation (optimal $D$) is a function of the tasks' covariance $WW^\top$.

Thus, we propose to solve the minimization problem

$$\inf \left\{ \text{Reg}(W, D) : W \in \mathbb{R}^{d \times T}, D \in \mathbf{S}^d_{++}, \text{tr}\, D \leq 1 \right\} \tag{2.4}$$

for functions $f$ belonging to an appropriate class. As we shall see in Section 4, the upper bound on the trace of $D$ in (2.4) prevents the infimum from being zero, which would lead to overfitting. Moreover, even though the infimum above is not attained in general, the problem in $W$ resulting after partial minimization over $D$ admits a minimizer.

Since the first term in (2.1) is independent of $D$, we can first optimize the second term with respect to $D$. That is, we can compute the infimum

$$\Omega_f(W) := \inf \left\{ \operatorname{tr}(F(D)WW^\top) : D \in \mathbf{S}_{++}^d, \operatorname{tr} D \leq 1 \right\}. \tag{2.5}$$

In this way we could end up with an optimization problem in $W$ only. However, in general this would be a complex matrix optimization problem. It may require sophisticated optimization tools such as semidefinite programming, which may not scale well with the size of $W$. Fortunately, as we shall show, problem (2.4) can be efficiently solved by alternately minimizing over $D$ and $W$. In particular, in Section 4 we shall show that $\Omega_f$ is a function of the singular values of $W$ only. Hence, the only matrix operation required by alternate minimization is singular value decomposition and the rest are merely vector problems.

Finally, we note that the ideas above may be extended naturally to a reproducing kernel Hilbert space setting [3].

## 3 Joint Convexity via Matrix Concave Functions

In this section, we address the issue of convexity of the regularization function (2.1). Our main result characterizes the class of spectral functions $F$ for which the term $w^\top F(D)w$ is jointly convex in $(w, D)$, which in turn implies that (2.4) is a convex optimization problem.

To illustrate our result, we require the matrix analytic concept of concavity, see, for example, [7]. We say that the real-valued function $g : (0, \infty) \to \mathbb{R}$ is *matrix concave of order $d$* if

$$\lambda G(A) + (1 - \lambda)G(B) \preceq G(\lambda A + (1 - \lambda)B) \qquad \forall A, B \in \mathbf{S}_{++}^d \text{ and } \lambda \in [0, 1],$$

where $G$ is defined as in (2.3). The notation $\preceq$ denotes the *Loewner partial order* on $\mathbf{S}^d$: $C \preceq D$ if and only if $D - C$ is positive semidefinite. If $g$ is a matrix concave function of order $d$ for any $d \in \mathbb{N}$, we simply say that $g$ is *matrix concave*. We also say that $g$ is matrix convex (of order $d$) if $-g$ is matrix concave (of order $d$). Clearly, matrix concavity implies matrix concavity of smaller orders (and hence standard concavity).

**Theorem 3.1.** *Let $F : \mathbf{S}_{++}^d \to \mathbf{S}_{++}^d$ be a spectral function. Then the function $\rho : \mathbb{R}^d \times \mathbf{S}_{++}^d \to [0, \infty)$ defined as $\rho(w, D) = w^\top F(D)w$ is jointly convex if and only if $\frac{1}{f}$ is matrix concave of order $d$.*

**Proof.** By definition, $\rho$ is convex if and only if, for any $w_1, w_2 \in \mathbb{R}^d, D_1, D_2 \in \mathbf{S}_{++}^d$ and $\lambda \in (0, 1)$, it holds that

$$\rho(\lambda w_1 + (1 - \lambda)w_2, \lambda D_1 + (1 - \lambda)D_2) \leq \lambda \rho(w_1, D_1) + (1 - \lambda)\rho(w_2, D_2).$$

Let $C := F(\lambda D_1 + (1 - \lambda)D_2), A := F(D_1)/\lambda, B := F(D_2)/(1 - \lambda), w := \lambda w_1 + (1 - \lambda)w_2$ and $z := \lambda w_1$. Using this notation, the above inequality can be rewritten as

$$w^\top C w \leq z^\top A z + (w - z)^\top B(w - z) \qquad \forall w, z \in \mathbb{R}^d. \tag{3.1}$$

The right hand side in (3.1) is minimized for $z = (A + B)^{-1}Bw$ and hence (3.1) is equivalent to

$$w^\top C w \leq w^\top \left[ B(A + B)^{-1}A(A + B)^{-1}B + \left( I - (A + B)^{-1}B \right)^\top B \left( I - (A + B)^{-1}B \right) \right] w,$$

$\forall\, w \in \mathbb{R}^d$, or to

$$\begin{aligned}
C &\preceq B(A + B)^{-1}A(A + B)^{-1}B + \left( I - (A + B)^{-1}B \right)^\top B \left( I - (A + B)^{-1}B \right) \\
&= B(A + B)^{-1}A(A + B)^{-1}B + B - 2B(A + B)^{-1}B + B(A + B)^{-1}B(A + B)^{-1}B \\
&= B - B(A + B)^{-1}B = (A^{-1} + B^{-1})^{-1},
\end{aligned}$$

where the last equality follows from the matrix inversion lemma [10, Sec. 0.7]. The above inequality is identical to (see e.g. [10, Sec. 7.7])

$$A^{-1} + B^{-1} \preceq C^{-1},$$

or, using the initial notation,

$$\lambda\big(F(D_1)\big)^{-1} + (1-\lambda)\big(F(D_2)\big)^{-1} \preceq \big(F(\lambda D_1 + (1-\lambda)D_2)\big)^{-1}.$$

By definition, this inequality holds for any $D_1, D_2 \in \mathbf{S}_{++}^d, \lambda \in (0,1)$ if and only if $\frac{1}{f}$ is matrix concave of order $d$. □

Examples of matrix concave functions on $(0,\infty)$ are $\log(x+1)$ and the function $x^s$ for $s \in [0,1]$ – see [7] for other examples and theoretical results. We conclude with the remark that, whenever $\frac{1}{f}$ is matrix concave of order $d$, function $\Omega_f$ in (2.5) is convex, because it is the partial infimum of a jointly convex function [9, Sec. IV.2.4].

## 4 Regularization with Schatten $L_p$ Prenorms

### 4.1 Partial Minimization of the Penalty Term

In this section, we focus on the family of negative power functions $f$ and obtain that function $\Omega_f$ in (2.5) relates to the Schatten $L_p$ prenorms. We start by showing that problem (2.5) reduces to a minimization problem in $\mathbb{R}^d$, by application of a useful matrix inequality. In the following, we let $B$ take the place of $WW^\top$ for brevity.

**Lemma 4.1.** *Let $F : \mathbf{S}^d \to \mathbf{S}^d$ be a spectral function, $B \in \mathbf{S}^d$ and $\beta_i$, $i \in \mathbb{N}_d$, the eigenvalues of $B$. Then,*

$$\inf\{\operatorname{tr}(F(D)B) : D \in \mathbf{S}_{++}^d, \operatorname{tr} D \le 1\} = \inf\left\{\sum_{i \in \mathbb{N}_d} f(\delta_i)\beta_i \ : \ \delta_i > 0, i \in \mathbb{N}_d, \sum_{i \in \mathbb{N}_d} \delta_i \le 1\right\}.$$

*Moreover, for the infimum on the left to be attained, $F(D)$ has to share a set of eigenvectors with $B$ so that the corresponding eigenvalues are in the reverse order as the $\beta_i$.*

**Proof.** We use an inequality of Von Neumann [13, Sec. H.1.h] to obtain, for all $X, Y \in \mathbf{S}^d$, that

$$\operatorname{tr}(XY) \ge \sum_{i \in \mathbb{N}_d} \lambda_i \mu_i$$

where $\lambda_i$ and $\mu_i$ are the eigenvalues of $X$ and $Y$ in nonincreasing and nondecreasing order, respectively. The equality is attained whenever $X = U\operatorname{Diag}(\lambda)U^\top, Y = U\operatorname{Diag}(\mu)U^\top$ for some $U \in \mathbf{O}^d$. Applying this inequality for $X = F(D), Y = B$ and denoting $f(\delta_i) = \lambda_i, i \in \mathbb{N}_d$, the result follows. □

Using this lemma, we can now derive the solution of problem (2.5) in the case that $f$ is a negative power function.

**Proposition 4.2.** *Let $B \in \mathbf{S}_+^d$ and $s \in (0,1]$. Then we have that*

$$(\operatorname{tr} B^s)^{\frac{1}{s}} = \inf\left\{\operatorname{tr}(D^{\frac{s-1}{s}}B) : D \in \mathbf{S}_{++}^d, \operatorname{tr} D \le 1\right\}.$$

*Moreover, if $B \in \mathbf{S}_{++}^d$ the infimum is attained and the minimizer is given by $D = \dfrac{B^s}{\operatorname{tr} B^s}$.*

**Proof.** By Lemma 4.1, it suffices to show the analogous statement for vectors, namely that

$$\left(\sum_{i \in \mathbb{N}_d} \beta_i^s\right)^{\frac{1}{s}} = \inf\left\{\sum_{i \in \mathbb{N}_d} \delta_i^{\frac{s-1}{s}}\beta_i : \delta_i > 0, i \in \mathbb{N}_d, \sum_{i \in \mathbb{N}_d} \delta_i \le 1\right\}$$

where $\beta_i \ge 0, i \in \mathbb{N}_d$. To this end, we apply Hölder's inequality with $p = \frac{1}{s}$ and $q = \frac{1}{1-s}$ :

$$\sum_{i \in \mathbb{N}_d} \beta_i^s = \sum_{i \in \mathbb{N}_d} \left(\delta_i^{\frac{s-1}{s}}\beta_i\right)^s \delta_i^{1-s} \le \left(\sum_{i \in \mathbb{N}_d} \delta_i^{\frac{s-1}{s}}\beta_i\right)^s \left(\sum_{i \in \mathbb{N}_d} \delta_i\right)^{1-s} \le \left(\sum_{i \in \mathbb{N}_d} \delta_i^{\frac{s-1}{s}}\beta_i\right)^s.$$

When $\beta_i > 0, i \in \mathbb{N}_d$, the equality is attained for $\delta_i = \dfrac{\beta_i^s}{\sum_{j \in \mathbb{N}_d} \beta_j^s}, i \in \mathbb{N}_d$. To show that the inequality is sharp in all other cases, we replace $\beta_i$ by $\beta_{i,\varepsilon} := \beta_i + \varepsilon$, $i \in \mathbb{N}_d, \varepsilon > 0$, define $\delta_{i,\varepsilon} = \beta_{i,\varepsilon}^s / (\sum_j \beta_{j,\varepsilon}^s)$ and take the limits as $\varepsilon \to 0$. □

The above result implies that the regularization problem (2.4) is conceptually equivalent to regularization with a Schatten $L_p$ prenorm of $W$, when the coupling function $f$ takes the form $f(x) = x^{1-\frac{2}{p}}$ with $p \in (0, 2]$, $p = 2s$. The Schatten $L_p$ prenorm is the $L_p$ prenorm of the singular values of a matrix. In particular, trace norm regularization (see [1, 17]) corresponds to the case $p = 1$. We also note that generalization error bounds for Schatten $L_p$ norm regularization can be derived along the lines of [14].

## 4.2 Learning Algorithm

Lemma 4.1 demonstrates that optimization problems such as (2.4) with spectral regularizers of the form (2.2) are computationally appealing, since they decompose to vector problems in $d$ variables along with singular value decomposition of the matrix $W$. In particular, for the Schatten $L_p$ prenorm with $p \in (0, 2]$, the proof of Proposition 4.2 suggests a way to solve problem (2.4). We modify the penalty term (2.2) as

$$\text{Penalty}_\varepsilon(W, D) = \text{tr}\big(F(D)(WW^\top + \varepsilon I)\big), \tag{4.1}$$

where $\varepsilon > 0$ and let $\text{Reg}_\varepsilon(W, D) = \text{Err}(W) + \gamma \, \text{Penalty}_\varepsilon(W, D)$ be the corresponding regularization function. By Proposition 4.2, for a fixed $W \in \mathbb{R}^{d \times T}$ there is a unique minimizer of $\text{Penalty}_\varepsilon$ (under the constraints in (2.5)), given by the formula

$$D_\varepsilon(W) = \frac{(WW^\top + \varepsilon I)^{\frac{p}{2}}}{\text{tr}(WW^\top + \varepsilon I)^{\frac{p}{2}}} \,. \tag{4.2}$$

Moreover, there exists a minimizer of problem (2.4), which is unique if $p \in (1, 2]$.

Therefore, we can solve problem (2.4) using an alternating minimization algorithm, which is an extension of the one presented in [4] for the special case $F(D) = D^{-1}$. Each iteration of the algorithm consists of two steps. In the first step, we keep $D$ fixed and minimize over $W$. This consists in solving the problem

$$\min \left\{ \sum_{t \in \mathbb{N}_T} L_t(w_t) + \gamma \sum_{t \in \mathbb{N}_T} w_t^\top F(D) w_t \; : \; W \in \mathbb{R}^{d \times T} \right\}.$$

This minimization can be carried out independently for each task since the regularizer decouples when $D$ is fixed. Specifically, introducing new variables for $(F(D))^{\frac{1}{2}} w_t$ yields a standard $L_2$ regularization problem for each task with the *same* kernel $K(x, z) = x^\top (F(D))^{-1} z$, $x, z \in \mathbb{R}^d$. In other words, we simply learn the parameters $w_t$ – the columns of matrix $W$ – independently by a regularization method, for example by an SVM or ridge regression method, for which there are well developed tool boxes. In the second step, we keep matrix $W$ fixed and minimize over $D$ using equation (4.2).

Space limitations prevent us from providing a convergence proof of the algorithm. We only note that following the proof detailed in [3] for the case $p = 1$, one can show that the sequence produced by the algorithm converges to the unique minimizer of $\text{Reg}_\varepsilon$ if $p \in [1, 2]$, or to a local minimizer if $p \in (0, 1)$. Moreover, by [3, Thm. 3] as $\varepsilon$ goes to zero the algorithm converges to a solution of problem (2.4), if $p \in [1, 2]$. In theory, an algorithm without $\varepsilon$-perturbation does not converge to a minimizer, since the columns of $W$ and $D$ always remain in the initial column space. In practice, however, we have observed that even such an algorithm converges to an optimal solution, because of round-off effects.

## 5 Relation to Learning the Kernel

In this section, we discuss the connection between the multi-task framework (2.1)-(2.4) and the framework for learning the kernel, see [11, 15] and references therein. To this end, we define the kernel $K_f(D)(x, z) = x^\top (F(D))^{-1} z$, $x, z \in \mathbb{R}^d$, the set of kernels $\mathcal{K}_f = \{K_f(D) : D \in \mathbf{S}_{++}^d, \text{tr} D \leq 1\}$ and, for every kernel $K$, the task kernel matrix $K_t = (K(x_{ti}, x_{tj}) : i, j \in \mathbb{N}_m)$, $t \in \mathbb{N}_T$. It is easy to prove, using Weyl's monotonicity theorem [10, Sec. 4.3] and [7, Thm. V.2.5], that the set $\mathcal{K}_f$ is convex if and only if $\frac{1}{f}$ is matrix concave. By the well-known representer theorem (see e.g. [11]), problem (2.4) is equivalent to minimizing the function

$$\sum_{t \in \mathbb{N}_T} \left( \sum_{i \in \mathbb{N}_m} \ell(y_{ti}, (K_t c_t)_i) + \gamma \, c_t^\top K_t c_t \right) \tag{5.1}$$

over $c_t \in \mathbb{R}^m$ (for $t \in \mathbb{N}_T$) and $K \in \mathcal{K}_f$. It is apparent that the function (5.1) is not jointly convex in $c_t$ and $K$. However, minimizing each term over the vector $c_t$ gives a convex function of $K$.

**Proposition 5.1.** *Let $\mathcal{K}$ be the set of all reproducing kernels on $\mathbb{R}^d$. If $\ell(y, \cdot)$ is convex for any $y \in \mathbb{R}$ then the function $E_t : \mathcal{K} \to [0, \infty)$ defined for every $K \in \mathcal{K}$ as*

$$E_t(K) = \min \left\{ \sum_{i \in \mathbb{N}_m} \ell(y_{ti}, (K_t c)_i) + \gamma\, c^\top K_t c : c \in \mathbb{R}^m \right\}$$

*is convex.*

**Proof.** Without loss of generality, we can assume as in [15] that $K_t$ are invertible for all $t \in \mathbb{N}_T$. For every $a \in \mathbb{R}^m$ and $K \in \mathcal{K}$, we define the function $G_t(a, K) = \sum_{i \in \mathbb{N}_m} \ell(y_{ti}, a_i) + \gamma\, a^\top K_t^{-1} a$, which is jointly convex by Theorem 3.1. Clearly, $E_t(K) = \min\{G_t(a, K) : a \in \mathbb{R}^m\}$. Recalling that the partial minimum of a jointly convex function is convex [9, Sec. IV.2.4], we obtain the convexity of $E_t$. $\qquad\square$

The fact that the function $E_t$ is convex has already been proved in [15], using minimax theorems and Fenchel duality. Here, we were able to simplify the proof of this result by appealing to the joint convexity property stated in Theorem 3.1.

## 6 Experiments

In this section, we first report a comparison of the computational cost between the alternating minimization algorithm and the gradient descent algorithm. We then study how performance varies for different $L_p$ regularizers, compare our approach with other multi-task learning methods and report experiments on transfer learning.

We used two data sets in our experiments. The first one is the *computer survey data* from [12]. It was taken from a survey of 180 persons who rated the likelihood of purchasing one of 20 different personal computers. Here the persons correspond to tasks and the computer models to examples. The input represents 13 different computer characteristics (price, CPU, RAM etc.) while the output is an integer rating on the scale $0 - 10$. Following [12], we used the first 8 examples per task as the training data and the last 4 examples per task as the test data. We measured the root mean square error of the predicted from the actual ratings for the test data, averaged across people.

The second data set is the *school data set* from the Inner London Education Authority (see *http://www.cmm.bristol.ac.uk/learning-training/multilevel-m-support/datasets.shtml*). It consists of examination scores of 15362 students from 139 secondary schools in London. Thus, there are 139 tasks, corresponding to predicting student performance in each school. The input consists of the year of the examination, 4 school-specific and 3 student-specific attributes. Following [5], we replaced categorical attributes with binary ones, to obtain 27 attributes in total. We generated the training and test sets by 10 random splits of the data, so that 75% of the examples from each school (task) belong to the training set and 25% to the test set. Here, in order to compare our results with those in [5], we used the measure of percentage explained variance, which is defined as one minus the mean squared test error over the variance of the test data and indicates the percentage of variance explained by the prediction model. Finally, we note that in both data sets we used the square loss, tuned the regularization parameter $\gamma$ with 5-fold cross-validation and added an additional input component accounting for the bias term.

In the first experiment, we study the computational cost of the alternating minimization algorithm against the gradient descent algorithm, both implemented in Matlab, for the Schatten $L_{1.5}$ norm. The left plot in Figure 1 shows the value of the objective function (2.1) versus the number of iterations, on the computer survey data. The curves for different learning rates $\eta$ are shown, whereas for rates greater than $0.05$ gradient descent diverges. The alternating algorithm curve for $\varepsilon = 10^{-16}$ is also shown. We further note that for both data sets our algorithm typically needed less than 30 iterations to converge. The right plot depicts the CPU time (in seconds) needed to reach a value of the objective function which is less than $10^{-5}$ away from the minimum, versus the number of tasks. It is clear that our algorithm is at least an order of magnitude faster than gradient descent with the optimal learning rate and scales better with the number of tasks. We note that the computational cost of our method is mainly due to the $T$ ridge regressions in the supervised step (learning $W$) and the singular

value decomposition in the unsupervised step (learning $D$). A singular value decomposition is also needed in gradient descent, for computing the gradient of the Schatten $L_p$ norm. We have observed that the cost per iteration is smaller for gradient descent but the number of iterations is at least an order of magnitude larger, leading to the large difference in time cost.

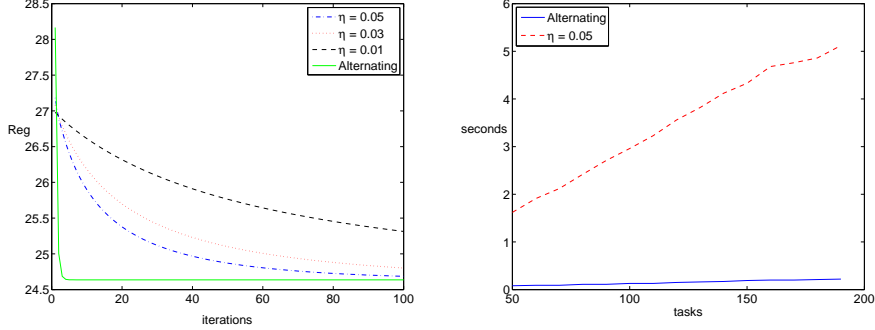

Figure 1: Comparison between the alternating algorithm and the gradient descent algorithm.

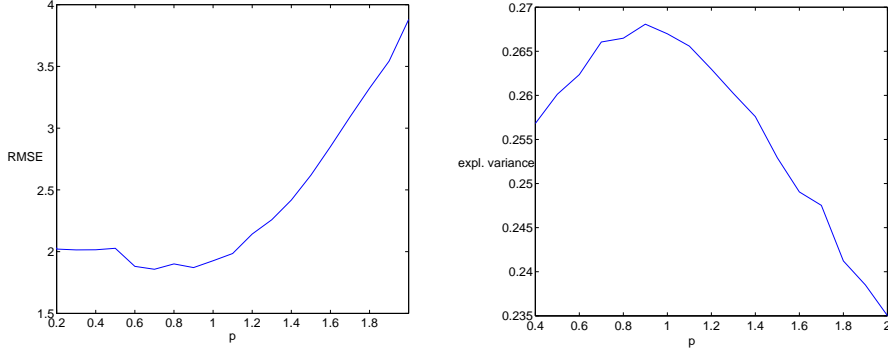

Figure 2: Performance versus $p$ for the computer survey data (left) and the school data (right).

Table 1: Comparison of different methods on the computer survey data (left) and school data (right).

| Method | RMSE |
|---|---|
| p = 2 | 3.88 |
| p = 1 | 1.93 |
| p = 0.7 | 1.86 |
| Hierarchical Bayes [12] | 1.90 |

| Method | Explained variance |
|---|---|
| p = 2 | $23.5 \pm 2.0\%$ |
| p = 1 | $26.7 \pm 2.0\%$ |
| Hierarchical Bayes [5] | $29.5 \pm 0.4\%$ |

In the second experiment we study the statistical performance of our method as the spectral function changes. Specifically, we choose functions giving rise to Schatten $L_p$ prenorms, as discussed in Section 4. The results, shown in Figure 2, indicate that the trace norm is the best *norm* on these data sets. However, on the computer survey data a value of $p$ less than one gives the best result overall. From this we speculate that our method can even approximate well the solutions of certain non-convex problems. In contrast, on the school data the trace norm gives almost the best result.

Next, in Table 1, we compare our algorithm with the hierarchical Bayes (HB) method described in [5, 12]. This method also learns a matrix $D$ using Bayesian inference. Our method improves on the HB method on the computer survey data and is competitive on the school data (even though our regularizer is simpler than HB and the data splits of [5] are not available).

Finally, we present preliminary results on transfer learning. On the computer survey data, we trained our method with $p = 1$ on 150 randomly selected tasks and then used the learned structure matrix $D$ for training 30 ridge regressions on the remaining tasks. We obtained an RMSE of 1.98 on these 30 "new" tasks, which is not much worse than an RMSE of 1.88 on the 150 tasks. In comparison, when

using the raw data ($D = \frac{I}{d}$) on the 30 tasks we obtained an RMSE of 3.83. A similar experiment was performed on the school data, first training on a random subset of 110 schools and then transferring $D$ to the remaining 29 schools. We obtained an explained variance of $19.2\%$ on the new tasks. This was worse than the explained variance of $24.8\%$ on the 110 tasks but still better than the explained variance of $13.9\%$ with the raw representation.

## 7 Conclusion

We have presented a spectral regularization framework for learning the structure shared by many supervised tasks. This structure is summarized by a positive definite matrix which is a spectral function of the tasks' covariance matrix. The framework is appealing both theoretically and practically. Theoretically, it brings to bear the rich class of spectral functions which is well-studied in matrix analysis. Practically, we have argued via the concrete example of negative power spectral functions, that the tasks' parameters and the structure matrix can be efficiently computed using an alternating minimization algorithm, improving upon state of the art statistical performance on two real data sets. A natural question is to which extent the framework can be generalized to allow for more complex task sharing mechanisms, in which the structure parameters depend on higher order statistical properties of the tasks.

## Acknowledgements

This work was supported by EPSRC Grant EP/D052807/1, NSF Grant DMS 0712827 and by the IST Programme of the European Commission, PASCAL Network of Excellence IST-2002-506778.

## References

[1] J. Abernethy, F. Bach, T. Evgeniou, and J-P. Vert. Low-rank matrix factorization with attributes. Technical Report N24/06/MM, Ecole des Mines de Paris, 2006.

[2] R. K. Ando and T. Zhang. A framework for learning predictive structures from multiple tasks and unlabeled data. *Journal of Machine Learning Research*, 6:1817–1853, 2005.

[3] A. Argyriou, T. Evgeniou, and M. Pontil. Convex multi-task feature learning. *Machine Learning*, 2007. In press.

[4] A. Argyriou, T. Evgeniou, and M. Pontil. Multi-task feature learning. In *Advances in Neural Information Processing Systems 19*, pages 41–48. 2007.

[5] B. Bakker and T. Heskes. Task clustering and gating for bayesian multi–task learning. *Journal of Machine Learning Research*, 4:83–99, 2003.

[6] J. Baxter. A model for inductive bias learning. *J. of Artificial Intelligence Research*, 12:149–198, 2000.

[7] R. Bhatia. *Matrix Analysis*. Graduate texts in Mathematics. Springer, 1997.

[8] R. Chari, W.W. Lockwood, and B.P. Coe et al. Sigma: a system for integrative genomic microarray analysis of cancer genomes. *BMC Genomics*, 7:324, 2006.

[9] J.-B. Hiriart-Urruty and C. Lemaréchal. *Convex Analysis and Minimization Algorithms*. Springer, 1996.

[10] R. A. Horn and C. R. Johnson. *Matrix Analysis*. Cambridge University Press, 1985.

[11] G.R.G. Lanckriet, N. Cristianini, P. Bartlett, L. El Ghaoui, and M.I. Jordan. Learning the kernel matrix with semidefinite programming. *Journal of Machine Learning Research*, 5:27–72, 2005.

[12] P. J. Lenk, W. S. DeSarbo, P. E. Green, and M. R. Young. Hierarchical Bayes conjoint analysis: recovery of partworth heterogeneity from reduced experimental designs. *Marketing Science*, 15(2):173–191, 1996.

[13] A. W. Marshall and I. Olkin. *Inequalities: Theory of Majorization and its Applications*. Academic Press, 1979.

[14] A. Maurer. Bounds for linear multi-task learning. *J. of Machine Learning Research*, 7:117–139, 2006.

[15] C.A. Micchelli and M. Pontil. Learning the kernel function via regularization. *Journal of Machine Learning Research*, 6:1099–1125, 2005.

[16] R. Raina, A. Y. Ng, and D. Koller. Constructing informative priors using transfer learning. In *Proceedings of the 23rd International Conference on Machine Learning*, 2006.

[17] N. Srebro, J. D. M. Rennie, and T. S. Jaakkola. Maximum-margin matrix factorization. In *Advances in Neural Information Processing Systems 17*, pages 1329–1336. 2005.

[18] A. Torralba, K. P. Murphy, and W. T. Freeman. Sharing features: efficient boosting procedures for multiclass object detection. In *Proc. of Conf. on Computer Vision and Pattern Recognition*. 2:762-769, 2004.

[19] J. Zhang, Z. Ghahramani, and Y. Yang. Learning multiple related tasks using latent independent component analysis. In *Advances in Neural Information Processing Systems 18*, pages 1585–1592. 2006.

